# No Unbiased Estimator of the Variance of K-Fold Cross-Validation

**Yoshua Bengio and Yves Grandvalet**
Dept. IRO, Université de Montréal
C.P. 6128, Montreal, Qc, H3C 3J7, Canada
{bengioy,grandvay}@iro.umontreal.ca

## Abstract

Most machine learning researchers perform quantitative experiments to estimate generalization error and compare algorithm performances. In order to draw statistically convincing conclusions, it is important to estimate the uncertainty of such estimates. This paper studies the estimation of uncertainty around the K-fold cross-validation estimator. The main theorem shows that there exists no universal unbiased estimator of the variance of K-fold cross-validation. An analysis based on the eigendecomposition of the covariance matrix of errors helps to better understand the nature of the problem and shows that naive estimators may grossly underestimate variance, as confirmed by numerical experiments.

## 1 Introduction

The standard measure of accuracy for trained models is the prediction error (PE), i.e. the expected loss on future examples. Learning algorithms themselves are often compared on their average performance, which estimates expected value of prediction error (EPE) over training sets. If the amount of data is large enough, PE can be estimated by the mean error over a hold-out test set. The hold-out technique does not account for the variance with respect to the training set, and may thus be considered inappropriate for the purpose of algorithm comparison [4]. Moreover, it makes an inefficient use of data which forbids its application to small sample sizes. In this situation, one resorts to computer intensive resampling methods such as cross-validation or bootstrap to estimate PE or EPE. We focus here on K-fold cross-validation. While it is known that cross-validation provides an unbiased estimate of EPE, it is also known that its variance may be very large [2]. This variance should be estimated to provide faithful confidence intervals on PE or EPE, and to test the significance of observed differences between algorithms. This paper provides theoretical arguments showing the difficulty of this estimation.

The difficulties of the variance estimation have already been addressed [4, 7, 8]. Some distribution-free bounds on the deviations of cross-validation are available, but they are specific to locally defined classifiers, such as nearest neighbors [3]. This paper builds upon the work of Nadeau and Bengio [8], which investigated in detail the theoretical and practical merits of several estimators of the variance of cross-validation. Our analysis departs from this work in the sampling procedure defining the cross-validation estimate. While [8] considers K independent training and test splits, we focus on the standard K-fold cross-

validation procedure, with no overlap between test sets: each example is used once and only once as a test example.

## 2   General Framework

Formally, we have a training set $D = \{\mathbf{z}_1, \ldots, \mathbf{z}_n\}$, with $\mathbf{z}_i \in \mathcal{Z}$, assumed independently sampled from an unknown distribution $P$. We also have a learning algorithm $A : \mathcal{Z}^* \to \mathcal{F}$ which maps a data set to a function. Here we consider symmetric algorithms, i.e. $A$ is insensitive to the ordering of examples in the training set $D$. The discrepancy between the prediction and the observation $\mathbf{z}$ is measured by a loss functional $L : \mathcal{F} \times \mathcal{Z} \to \mathbb{R}$. For example one may take in regression $L(f, (\mathbf{x}, y)) = (f(\mathbf{x}) - y)^2$, and in classification $L(f, (\mathbf{x}, y)) = 1_{f(\mathbf{x}) \neq y}$.

Let $f = A(D)$ be the function returned by algorithm $A$ on the training set $D$. In application-based evaluation, the goal of learning is usually stated as the minimization of the expected loss of $f = A(D)$ on future test examples:

$$\mathrm{PE}(D) = E[L(f, \mathbf{z})] \ , \tag{1}$$

where the expectation is taken with respect to $\mathbf{z} \sim P$. To evaluate and compare learning algorithms [4] we care about the expected performance of learning algorithm $A$ over different training sets:

$$\mathrm{EPE}(n) = E[L(A(D), \mathbf{z})] \ , \tag{2}$$

where the expectation is taken with respect to $D \times \mathbf{z}$ independently sampled from $P^n \times P$.

When $P$ is unknown, PE and EPE have to be estimated, and it is crucial to assess the uncertainty attached to this estimation. Although this point is often overlooked, estimating the variance of the estimates $\widehat{\mathrm{PE}}$ and $\widehat{\mathrm{EPE}}$ requires caution, as illustrated here.

### 2.1   Hold-out estimates of performance

The mean error over a hold-out test set estimates PE, and the variance of $\widehat{\mathrm{PE}}$ is given by the usual variance estimate for means of independent variables. However, this variance estimator is not suited to $\widehat{\mathrm{EPE}}$: the test errors are correlated when the training set is considered as a random variable.

Figure 1 illustrates how crucial it is to take these correlations into account. The average ratio (estimator of variance/empirical variance) is displayed for two variance estimators, in an ideal situation where 10 independent training and test sets are available. The average of $\widehat{\theta}_1/\theta$, the naive variance estimator ignoring correlations, shows that this estimate is highly down-biased, even for large sample sizes.

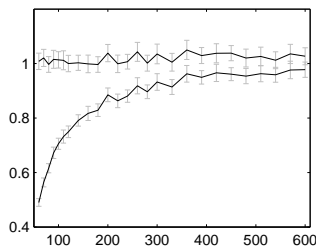

Figure 1: Average ratio (estimator of variance/empirical variance) on 100 000 experiments: $\widehat{\theta}_1/\theta$ (ignoring correlations, lower curve) and $\widehat{\theta}_2/\theta$ (taking into account correlations, upper curve) *vs.* sample size $n$. The error bars represent $\pm 2$ standard errors on the average value.

**Experiment 1** *Ideal hold-out estimate of* EPE.
*We have $K = 10$ independent training sets $D_1, \ldots, D_K$ of $n$ independent examples $\mathbf{z}_i = (\mathbf{x}_i, y_i)$, where $\mathbf{x_i} = (x_{i1}, \ldots, x_{id})'$ is a $d$-dimensional centered, unit covariance Gaussian variable ($d = 30$), $y_i = \sqrt{3/d} \sum_{k=1}^{d} x_{ik} + \varepsilon_i$ with $\varepsilon_i$ being independent, centered, unit variance Gaussian variables (the $\sqrt{3/d}$ factor provides $R^2 \simeq 3/4$). We also have $K$ independent test sets $T_1, \ldots, T_K$ of size $n$ sampled from the same distribution.
The learning algorithm consists in fitting a line by ordinary least squares, and the estimate of* EPE *is the average quadratic loss on test examples* $\widehat{\mathrm{EPE}} = \bar{L} = \frac{1}{K} \sum_{k=1}^{K} \frac{1}{n} \sum_{\mathbf{z}_i \in T_k} L_{ki}$, where $L_{ki} = L(A(D_k), \mathbf{z}_i)$.
*The first estimate of variance of* $\widehat{\mathrm{EPE}}$ *is* $\widehat{\theta}_1 = \frac{1}{Kn(Kn-1)} \sum_{k=1}^{K} \sum_i (L_{ki} - \bar{L})^2$, *which is unbiased provided there is no correlation between test errors. The second estimate is* $\widehat{\theta}_2 = \frac{1}{K(K-1)n^2} \sum_{k=1}^{K} \sum_{i,j} (L_{ki} - \bar{L})(L_{kj} - \bar{L})$, *which estimates correlations.*

Note that Figure 1 suggests that the naive estimator of variance $\widehat{\theta}_1$ asymptotically converges to the true variance. This can be shown by taking advantage of the results in this paper, as long as the learning algorithm converges ($\mathrm{PE}(D) \overset{a.s.}{\to} \lim_{n \to \infty} \mathrm{EPE}(n)$), i.e. provided that the only source of variability of $\widehat{\mathrm{EPE}}$ is due to the finite test size.

## 2.2 K-fold cross-validation estimates of performance

In K-fold cross-validation [9], the data set $D$ is first chunked into $K$ disjoint subsets (or *blocks*) of the same size $m = n/K$ (to simplify the analysis below we assume that $n$ is a multiple of $K$). Let us write $T_k$ for the $k$-th such block, and $D_k$ the training set obtained by removing the elements in $T_k$ from $D$. The estimator is

$$\mathrm{CV} = \frac{1}{K} \sum_{k=1}^{K} \frac{1}{m} \sum_{\mathbf{z}_i \in T_k} L(A(D_k), \mathbf{z}_i) \ . \tag{3}$$

Under stability assumptions on $A$, CV estimates $\mathrm{PE}(D)$ at least as accurately as the training error [6]. However, as CV is an average of unbiased estimates of $\mathrm{PE}(D_1)$, $\ldots, \mathrm{PE}(D_K)$, a more general statement is that CV estimates unbiasedly $\mathrm{EPE}(n-m)$.

Note that the forthcoming analysis also applies to the version of cross-validation dedicated to comparing algorithms, using matched pairs

$$\Delta\mathrm{CV} = \frac{1}{K} \sum_{k=1}^{K} \frac{1}{m} \sum_{\mathbf{z}_i \in T_k} L(A_1(D_k), \mathbf{z}_i) - L(A_2(D_k), \mathbf{z}_i) \ ,$$

and to the delete-$m$ jackknife estimate of $\mathrm{PE}(D)$ debiasing the training error (see e.g. [5]):

$$\mathrm{JK} = \frac{1}{n} \sum_{i=1}^{n} L(A(D), \mathbf{z}_i) - (K-1) \left( \frac{1}{K(n-m)} \sum_{k=1}^{K} \sum_{\mathbf{z}_i \in D_k} L(A(D_k), \mathbf{z}_i) - \frac{1}{n} \sum_{i=1}^{n} L(A(D), \mathbf{z}_i) \right).$$

In what follows, CV, $\Delta$CV and JK will generically be denoted by $\hat{\mu}$:

$$\hat{\mu} = \frac{1}{n} \sum_{i=1}^{n} e_i = \frac{1}{K} \sum_{k=1}^{K} \frac{1}{m} \sum_{i \in T_k} e_i \ ,$$

where, slightly abusing notation, $i \in T_k$ means $\mathbf{z}_i \in T_k$ and

$$\forall i \in T_k, \ e_i = \begin{cases} L(A(D_k), \mathbf{z}_i) & \text{for } \hat{\mu} = \mathrm{CV} \ , \\ L(A_1(D_k), \mathbf{z}_i) - L(A_2(D_k), \mathbf{z}_i) & \text{for } \hat{\mu} = \Delta\mathrm{CV} \ , \\ KL(A(D), \mathbf{z}_i) - \sum_{\ell \neq k} L(A(D_\ell), \mathbf{z}_i) & \text{for } \hat{\mu} = \mathrm{JK} \ . \end{cases}$$

Note that $\hat{\mu}$ is the average of identically distributed (dependent) variables. Thus, it asymptotically converges to a normally distributed variable, which is completely characterized by its expectation $E[\hat{\mu}]$ and its variance $\mathrm{Var}[\hat{\mu}]$.

## 3  Structure of the Covariance Matrix

The variance of $\hat{\mu}$ is $\theta = \frac{1}{n^2} \sum_{i,j} \mathrm{Cov}(e_i, e_j)$ . By using symmetry over permutations of the examples in $D$, we show that the covariance matrix has a simple block structure.

**Lemma 1** *Using the notation introduced in section 2.2, 1) all $e_i$ are identically distributed; 2) all pairs $(e_i, e_j)$ belonging to the same test block are jointly identically distributed; 3) all pairs $(e_i, e_j)$ belonging to different test blocks are jointly identically distributed;*

**Proof**: derived immediately from the permutation-invariance of $P(D)$ and the symmetry of $A$. See [1] for details and the proofs not shown here for lack of space.

**Corollary 1** *The covariance matrix $\boldsymbol{\Sigma}$ of cross-validation errors $\mathbf{e} = (e_1, \ldots, e_n)'$ has the simple block structure depicted in Figure 2: 1) all diagonal elements are identical $\forall i, \mathrm{Cov}(e_i, e_i) = \mathrm{Var}[e_i] = \sigma^2$; 2) all the off-diagonal entries of the $K$ $m \times m$ diagonal blocks are identical $\forall (i, j) \in T_k^2 : j \neq i, T(j) = T(i), \mathrm{Cov}(e_i, e_j) = \omega$; 3) all the remaining entries are identical $\forall i \in T_k, \forall j \in T_\ell : \ell \neq k, \mathrm{Cov}(e_i, e_j) = \gamma$.*

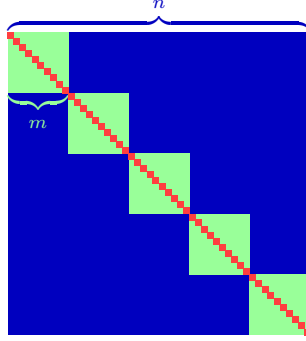

Figure 2: Structure of the covariance matrix.

**Corollary 2** *The variance of the cross-validation estimator is a linear combination of three moments:*

$$\theta = \frac{1}{n^2} \sum_{i,j} \mathrm{Cov}(e_i, e_j) = \frac{1}{n}\sigma^2 + \frac{m-1}{n}\omega + \frac{n-m}{n}\gamma \qquad (4)$$

Hence, the problem of estimating $\theta$ does not involve estimating $n(n+1)/2$ covariances, but it cannot be reduced to that of estimating a single variance parameter. Three components intervene, which may be interpreted as follows when $\hat{\mu}$ is the K-fold cross-validation estimate of EPE:

1. the variance $\sigma^2$ is the average (taken over training sets) variance of errors for "true" test examples (i.e. sampled independently from the training sets) when algorithm $A$ is fed with training sets of size $m(K-1)$;

2. the within-block covariance $\omega$ would also apply to these "true" test examples; it arises from the dependence of test errors stemming from the common training set.

3. the between-blocks covariance $\gamma$ is due to the dependence of training sets (which share $n(K-2)/K$ examples) and the fact that test block $T_k$ appears in all the training sets $D_\ell$ for $\ell \neq k$.

## 4 No Unbiased Estimator of $\text{Var}[\hat{\mu}]$ Exists

Consider a generic estimator $\hat{\theta}$ that depends on the sequence of cross-validation errors $\mathbf{e} = (e_1, e_2, \ldots, e_n)'$. Assuming $\hat{\theta}$ is analytic in $\mathbf{e}$, consider its Taylor expansion:

$$\hat{\theta} = \alpha_0 + \sum_i \alpha_1(i)e_i + \sum_{i,j} \alpha_2(i,j)e_ie_j + \sum_{i,j,k} \alpha_3(i,j,k)e_ie_je_k + \ldots \tag{5}$$

We first show that for unbiased variance estimates (i.e. $E[\hat{\theta}] = \text{Var}[\hat{\mu}]$), all the $\alpha_i$ coefficients must vanish except for the second order coefficients $\alpha_{2,i,j}$.

**Lemma 2** *There is no universal unbiased estimator of $\text{Var}[\hat{\mu}]$ that involves the $e_i$ in a non-quadratic way.*

**Proof***: Take the expected value of $\hat{\theta}$ expressed as in (5), and equate it with $\text{Var}[\hat{\mu}]$ (4).*

Since estimators that include moments other than the second moments in their expectation are biased, we now focus on estimators which are quadratic forms of the errors, i.e.

$$\hat{\theta} = \mathbf{e}'\mathbf{W}\mathbf{e} = \sum_{i,j} W_{ij}e_ie_j \ . \tag{6}$$

**Lemma 3** *The expectation of quadratic estimators $\hat{\theta}$ defined as in (6) is a linear combination of only three terms*

$$E[\hat{\theta}] = a(\sigma^2 + \mu^2) + b(\omega + \mu^2) + c(\gamma + \mu^2) \ , \tag{7}$$

*where $(a, b, c)$ are defined as follows:*

$$\begin{cases} a & \triangleq & \sum_{i=1}^n W_{ii} \ , \\ b & \triangleq & \sum_{k=1}^K \sum_{i \in T_k} \sum_{j \in T_k : j \neq i} W_{ij} \ , \\ c & \triangleq & \sum_{k=1}^K \sum_{\ell \neq k} \sum_{i \in T_k} \sum_{j \in T_\ell} W_{ij} \ . \end{cases}$$

*A "trivial" representer of estimators with this expected value is*

$$\hat{\theta} = as_1 + bs_2 + cs_3 \ , \tag{8}$$

*where $(s_1, s_2, s_3)$ are the only quadratic statistics of $\mathbf{e}$ that are invariants to the within blocks and between blocks permutations described in Lemma 1:*

$$\begin{cases} s_1 & \triangleq & \frac{1}{n}\sum_{i=1}^n e_i^2 \ , \\ s_2 & \triangleq & \frac{1}{n(m-1)} \sum_{k=1}^K \sum_{i \in T_k} \sum_{j \in T_k : j \neq i} e_ie_j \ , \\ s_3 & \triangleq & \frac{1}{n(n-m)} \sum_{k=1}^K \sum_{\ell \neq k} \sum_{i \in T_k} \sum_{j \in T_\ell} e_ie_j \ . \end{cases} \tag{9}$$

**Proof***: in (6), group the terms that have the same expected values (from Corollary 1).*

**Theorem 1** *There exists no universally unbiased estimator of $\text{Var}[\hat{\mu}]$.*

**Proof***: thanks to Lemma 2 and 3, it is enough to show that $E[\hat{\theta}] = \text{Var}[\hat{\mu}]$ has no solution for quadratic estimators:*

$$E[\hat{\theta}] = \text{Var}[\hat{\mu}] \Leftrightarrow a(\sigma^2 + \mu^2) + b(\omega + \mu^2) + c(\gamma + \mu^2) = \frac{1}{n}\sigma^2 + \frac{m-1}{n}\omega + \frac{n-m}{n}\gamma \ .$$

*Finding $(a, b, c)$ satisfying this equality for all admissible values of $(\mu, \sigma^2, \omega, \gamma)$ is impossible since it is equivalent to solving the following overdetermined system:*

$$\begin{cases} a & = & \frac{1}{n} \ , \\ b & = & \frac{m-1}{n} \ , \\ c & = & \frac{n-m}{n} \ , \\ a+b+c & = & 0 \end{cases} \tag{10}$$

*Q.E.D.*

## 5 Eigenanalysis of the covariance matrix

One way to gain insight on the origin of the negative statement of Theorem 1 is via the eigenanalysis of $\Sigma$, the covariance of $\mathbf{e}$. This decomposition can be performed analytically thanks to the very particular block structure displayed in Figure 2.

**Lemma 4** *Let $\mathbf{v}_k$ be the binary vector indicating the membership of each example to test block $k$. The eigenvalues of $\Sigma$ are as follows:*
- *$\lambda_1 = \sigma^2 - \omega$ with multiplicity $n - K$ and eigenspace orthogonal to $\{\mathbf{v}_k\}_{k=1}^K$;*
- *$\lambda_2 = \sigma^2 + (m-1)\omega - m\gamma$ with multiplicity $K - 1$ and eigenspace defined in the orthogonal of $\mathbf{1}$ by the basis $\{\mathbf{v}_k\}_{k=1}^K$;*
- *$\lambda_3 = \sigma^2 + (m-1)\omega + (n-m)\gamma$ with eigenvector $\mathbf{1}$.*

Lemma 4 states that the vector $\mathbf{e}$ can be decomposed into three uncorrelated parts: $n - K$ projections to the subspace orthogonal to $\{\mathbf{v}_k\}_{k=1}^K$, $K - 1$ projections to the subspace spanned by $\{\mathbf{v}_k\}_{k=1}^K$ in the orthogonal of $\mathbf{1}$, and one projection on $\mathbf{1}$.

A single vector example with $n$ independent elements can be seen as $n$ independent examples. Similarly, the uncorrelated projections of $\mathbf{e}$ can be equivalently represented by respectively $n - K$, $K - 1$ and one uncorrelated one-dimensional examples.

In particular, for the projection on $\mathbf{1}$, with a single example, the sample variance is null, resulting in the absence of unbiased variance estimator of $\lambda_3$. The projection of $\mathbf{e}$ on the eigenvector $\frac{1}{n}\mathbf{1}$ is precisely $\hat{\mu}$. Hence there is no unbiased estimate of $Var[\hat{\mu}] = \frac{\lambda_3}{n}$ when we have only one realization of the vector $\mathbf{e}$. For the same reason, even with simple parametric assumptions on $\mathbf{e}$ (such as $\mathbf{e}$ Gaussian), the maximum likelihood estimate of $\theta$ is not defined. Only $\lambda_1$ and $\lambda_2$ can be estimated unbiasedly. Note that this problem cannot be addressed by performing multiple K-fold splits of the data set. Such a procedure would not provide independent realizations of $\mathbf{e}$.

## 6 Possible values for $\omega$ and $\gamma$

Theorem 1 states that no estimator is unbiased, and in its demonstration, it is shown that the bias of any quadratic estimator is a linear combination of $\mu^2$, $\sigma^2$, $\omega$ and $\gamma$. Regarding estimation, it is thus interesting to see what constraints restrict their possible range.

**Lemma 5** *For $\hat{\mu} = \mathrm{CV}$ and $\hat{\mu} = \Delta\mathrm{CV}$, the following inequalities hold:*

$$\begin{cases} 0 & \leq \ \omega \ \leq & \sigma^2 \\ -\frac{1}{n-m}(\sigma^2 + (m-1)\omega) & \leq \ \gamma \ \leq & \frac{1}{m}(\sigma^2 + (m-1)\omega) \end{cases}$$
$$\Rightarrow \begin{cases} 0 & \leq \ \omega \ \leq \ \sigma^2 \\ -\frac{m}{n-m}\sigma^2 & \leq \ \gamma \ \leq \ \sigma^2 \ . \end{cases}$$

The admissible $(\omega, \gamma)$ region is very large, and there is no constraint linking $\mu$ to $\sigma^2$. Hence, we cannot propose a variance estimate with universally small bias.

## 7 Experiments

The bias of any quadratic estimator is a linear combination of $\mu^2$, $\sigma^2$, $\omega$ and $\gamma$. The admissible values provided earlier suggest that $\omega$ and $\gamma$ cannot be proved to be negligible compared to $\sigma^2$. This section illustrates that in practice, the contribution to the variance of $\hat{\mu}$ due to $\omega$ and $\gamma$ (see Equation (4)) can be of same order as the one due $\sigma^2$. This confirms that the estimators of $\theta$ should indeed take into account the correlations of $e_i$.

**Experiment 2** *True variance of K-fold cross-validation.*
*We repeat the experimental setup of Experiment 1, except that only one sample of size $n$ is available. Since cross-validation is known to be sensitive to the instability of algorithms,*

*in addition to this standard setup, we also consider another one with outliers:*
*The input* $\mathbf{x_i} = (x_{i1}, \ldots, x_{id})'$ *is still 30-dimensional, but it is now a mixture of two centered Gaussian: let* $t_i$ *be a binary variable, with* $P(t_i = 1) = p = 0.95$; $t_i = 1 \Rightarrow \mathbf{x_i} \sim \mathcal{N}(0, \mathbf{I})$, $t_i = 0 \Rightarrow \mathbf{x_i} \sim \mathcal{N}(0, 100\mathbf{I})$; $y_i = \sqrt{3/(d(p+100(1-p)))} \sum_{k=1}^{d} x_{ik} + \varepsilon_i$; $t_i = 1 \Rightarrow \varepsilon_i \sim \mathcal{N}(0, 1/(p+100(1-p)))$, $t_i = 0 \Rightarrow \varepsilon_i \sim \mathcal{N}(0, 100/(p+100(1-p)))$.

We now look at the variance of K-fold cross-validation ($K = 10$), and decompose in the three orthogonal components $\sigma^2$, $\omega$ and $\gamma$. The results are shown in Figure 3.

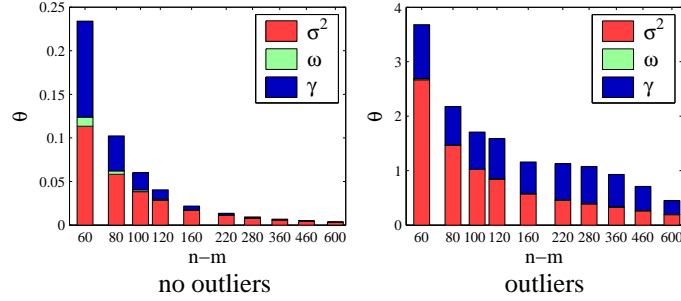

Figure 3: Contributions of $(\sigma^2, \omega, \gamma)$ to total variance $Var[CV]$ *vs.* $n - m$.

Without outliers, the contribution of $\gamma$ is very important for small sample sizes. For large sample sizes, the overall variance is considerably reduced and is mainly caused by $\sigma^2$ because the learning algorithm returns very similar answers for all training sets. When there are outliers, the contribution of $\gamma$ is of same order as the one of $\sigma^2$ even when the ratio of examples to free parameters is large (here up to 20). Thus, in difficult situations, where $A(D)$ varies according to the realization of $D$, neglecting the effect of $\omega$ and $\gamma$ can be expected to introduce a bias of the order of the true variance.

It is also interesting to see how these quantities are affected by the number of folds $K$. The decomposition of $\theta$ in $\sigma^2$, $\omega$ and $\gamma$ (4) does not imply that $K$ should be set either to $n$ or to 2 (according to the sign of $\omega - \gamma$) in order to minimize the variance of $\hat{\mu}$. Modifying $K$ affects $\sigma^2$, $\omega$ and $\gamma$ through the size and overlaps of the training sets $D_1, \ldots, D_K$, as illustrated in Figure 4. For a fixed sample size, the variance of $\hat{\mu}$ and the contribution of $\sigma^2$, $\omega$ and $\gamma$ vary smoothly with $K$ (of course, the mean of $\hat{\mu}$ is also affected in the process).

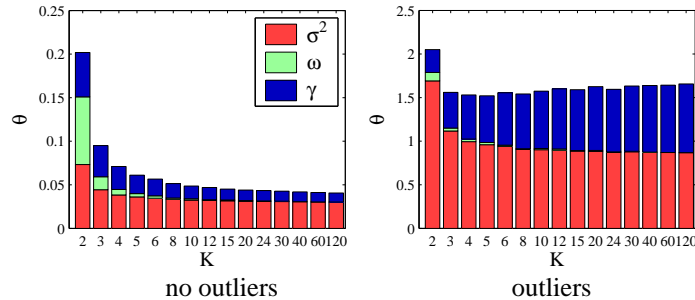

Figure 4: Contributions of $(\sigma^2, \omega, \gamma)$ to total variance $Var[CV]$ *vs.* $K$ for $n = 120$.

## 8 Discussion

The analysis presented in this paper for K-fold cross-validation can be instantiated to several interesting cases. First, when having $K$ *independent training and test sets* ($K = 1$

is the realistic case), the structure of **hold-out errors** resemble the one of cross-validation errors, with $\gamma = 0$. Knowing that allows to build the unbiased estimate $\widehat{\theta}_2$ given in 2.1: knowing that $\gamma = 0$ removes the third equation of system (10) in the proof of Theorem 1.

**Two-fold cross-validation** has been advocated to perform hypothesis testing [4]. It is a special case of K-fold cross-validation where the training blocks are mutually independent since they do not overlap. However, this independence does not modify the structure of **e** in the sense that $\gamma$ is not null. The between-block correlation stems from the fact that the training block $D_1$ is the test block $T_2$ and vice-versa.

Finally, **Leave-one-out cross validation** is another particular case, with $K = n$. The structure of the covariance matrix is simplified, without diagonal blocks. The estimation difficulties however remain: even in this particular case, there is no unbiased estimate of variance. From the definition of $b$ in Lemma 3, we have $b = 0$, and with $m = 1$ the linear system (10) still admits no solution.

To summarize, it is known that K-fold cross-validation may suffer from high variability, which can be responsible for bad choices in model selection and erratic behavior in the estimated expected prediction error [2, 4, 8]. This paper demonstrates that estimating the variance of K-fold cross-validation is difficult. Not only there is no unbiased estimate of this variance, but we have no theoretical result showing that this bias should be negligible in the non-asymptotical regime. The eigenanalysis of the covariance matrix of errors traces the problem back to the dependencies between test-block errors, which induce the absence of redundant pieces of information regarding the average test error. i.e. the K-fold cross-validation estimate. It is clear that this absence of redundancy is bound to provide difficulties in the estimation of variance.

Our experiments show that the bias incurred by ignoring test errors dependencies can be of the order of the variance itself, even for large sample sizes. Thus, the assessment of the significance of observed differences in cross-validation scores should be treated with much caution. The next step of this study consists in building and comparing variance estimators dedicated to the very specific structure of the test-block error dependencies.

## References

[1] Y. Bengio and Y. Grandvalet. No unbiased estimator of the variance of K-fold cross-validation. *Journal of Machine Learning Research*, 2003.

[2] L. Breiman. Heuristics of instability and stabilization in model selection. *The Annals of Statistics*, 24(6):2350–2383, 1996.

[3] L. Devroye, L. Györfi, and G. Lugosi. *A Probabilistic Theory of Pattern Recognition*. Springer, 1996.

[4] T. G. Dietterich. Approximate statistical tests for comparing supervised classification learning algorithms. *Neural Computation*, 10(7):1895–1924, 1999.

[5] B. Efron and R. J. Tibshirani. *An Introduction to the Bootstrap*, volume 57 of *Monographs on Statistics and Applied Probability*. Chapman & Hall, 1993.

[6] M. Kearns and D. Ron. Algorithmic stability and sanity-check bounds for leave-one-out cross-validation. *Neural Computation*, 11(6):1427–1453, 1996.

[7] R. Kohavi. A study of cross-validation and bootstrap for accuracy estimation and model selection. In *Proceedings of the Fourteenth International Joint Conference on Artificial Intelligence*, pages 1137–1143, 1995.

[8] C. Nadeau and Y. Bengio. Inference for the generalization error. *Machine Learning*, 52(3):239–281, 2003.

[9] M. Stone. Cross-validatory choice and assessment of statistical predictions. *Journal of the Royal Statistical Society, B*, 36(1):111–147, 1974.
